# Analog Neural Networks as Decoders

Ruth Erlanson*
Dept. of Electrical Engineering
California Institute of Technology
Pasadena, CA 91125

Yaser Abu-Mostafa
Dept. of Electrical Engineering
California Institute of Technology
Pasadena, CA 91125

## Abstract

Analog neural networks with feedback can be used to implement K-Winner-Take-All (KWTA) networks. In turn, KWTA networks can be used as decoders of a class of nonlinear error-correcting codes. By interconnecting such KWTA networks, we can construct decoders capable of decoding more powerful codes. We consider several families of interconnected KWTA networks, analyze their performance in terms of coding theory metrics, and consider the feasibility of embedding such networks in VLSI technologies.

## 1 INTRODUCTION: THE K-WINNER-TAKE-ALL NETWORK

We have previously demonstrated the use of a continuous Hopfield neural network as a K-Winner-Take-All (KWTA) network [Majani et al., 1989, Erlanson and Abu-Mostafa, 1988]. Given an input of $N$ real numbers, such a network will converge to a vector of $K$ positive one components and $(N - K)$ negative one components, with the positive positions indicating the $K$ largest input components. In addition, we have shown that the $\binom{N}{K}$ such vectors are the only stable states of the system.

One application of the KWTA network is the analog decoding of error-correcting codes [Majani et al., 1989, Platt and Hopfield, 1986]. Here, a known set of vectors (the codewords) are transmitted over a noisy channel. At the receiver's end of the channel, the initial vector must be reconstructed from the noisy vector.

If we select our codewords to be the $\binom{N}{K}$ vectors with $K$ positive one components and $(N-K)$ negative one components, then the KWTA neural network will perform this decoding task. Furthermore, the network decodes from the noisy analog vector to a binary codeword (so no information is lost in quantization of the noisy vector). Also, we have shown [Majani et al., 1989] that the KWTA network will perform the optimal decoding, maximum likelihood decoding (MLD), if we assume noise where the probability of a large noise spike is less than the probability of a small noise spike (such as additive white Gaussian noise). For this type of noise, an MLD outputs the codeword closest to the noisy received vector. Hence, the most straightforward implementation of MLD would involve the comparison of the noisy vector to all the codewords. For large codes, this method is computationally impractical.

Two important parameters of any code are its rate and minimum distance. The rate, or amount of information transmitted per bit sent over the channel, of this code is good (asymptotically approaches 1). The minimum distance of a code is the Hamming distance between the two closest codewords in the code. The minimum distance determines the error-correcting capabilities of a code. The minimum distance of the KWTA code is 2.

In our previous work, we have found that the KWTA network performs optimal decoding of a nonlinear code. However, the small minimum distance of this code limited the system's usefulness.

## 2    INTERCONNECTED KWTA NETWORKS

In order to look for more useful code-decoder pairs, we have considered interconnected KWTA networks. We have found two interesting families of codes:

### 2.1    THE HYPERCUBE FAMILY

A decoder for this family of codes has $m = n^i$ nodes. We label the nodes $x_1, x_2, \ldots x_i$ with $x_j \in 1, 2 \ldots n$. KWTA constraints are placed on sets of $n$ nodes which differ in only one index. For example, $\{1, 1, 1, \ldots, 1\}$, $\{2, 1, 1, \ldots, 1\}$, $\{3, 1, 1, \ldots, 1\}$, $\ldots$, $\{n, 1, 1, \ldots, 1\}$ are the nodes in one KWTA constraint.

For a two-dimensional system ($i = 2$) the nodes can be laid out in an array where the KWTA constraints will be along the rows and columns of the array. For the code associated with the two-dimensional system, we find that

$$\text{rate} \geq 1 - \frac{3 \log n}{2n}.$$

The minimum distance of this code is 4. Experimental results show that the decoder is nearly optimal.

In general, for an $i$-dimensional code, the minimum distance is $2^i$. The rate of these codes can be bounded only very roughly.

We also consider implementing these decoders on an integrated circuit. Because of the high level of interconnectivity of these decoders and the simple processing required at each node (or neuron) we assume that the interconnections will dictate the chip's size. Using a standard model for VLSI area complexity, we determine

that the circuit area scales as the square of the network size. Feature sizes of current mainstream technologies suggest that we could construct systems with $22^2 = 484$ (2-dimensional), $6^3 = 216$ (3-dimensional) and $5^4 = 625$ (4-dimensional) nodes. Thus, nontrivial systems could be constructed with current VLSI technology.

## 2.2   NET-GENERATED CODES

This family uses combinatorial nets to specify the nodes in the KWTA constraints. A net on $n^2$ points consists of parallel classes: Each class partitions the $n^2$ points into $n$ disjoint lines each containing $n$ points. Two lines from different classes intersect at exactly one point.

If we impose a KWTA constraint on the points on a line, a net can be used to generate a family of code-decoder pairs. If $n$ is the integer power of a prime number, we can use a projective plane to generate a net with $(n+1)$ classes. For example, in Table 1 we have the projective plane of order 2 ($n = 2$). A projective plane has $n^2 + n + 1$ points and $n^2 + n + 1$ lines where each line has $n + 1$ points and any 2 lines intersect in exactly one point.

Table 1: Projective Plane of Order 2. Points are numbered for clarity.

|  | points: | | | | | | |
|---|---|---|---|---|---|---|---|
|  | 1 | 2 | 3 | 4 | 5 | 6 | 7 |
| lines: | 1 | 1 |  | 1 |  |  |  |
|  |  | 1 | 1 |  | 1 |  |  |
|  |  |  | 1 | 1 |  | 1 |  |
|  |  |  |  | 1 | 1 |  | 1 |
|  | 1 |  |  |  | 1 | 1 |  |
|  |  | 1 |  |  |  | 1 | 1 |
|  | 1 |  | 1 |  |  |  | 1 |

We can generate a net of 3 (i.e., $n+1$) classes in the following way: Pick one line of the projective plane. Without loss of generality, we select the first line. Eliminate the points in that line from all the lines in the projective plane, as shown in Table 2. Renumber the remaining $n^2 + n + 1 - (n+1) = n^2$ points. These are the points of the net. The first class of the net is composed of the reduced lines which previously contained the first point (old label 1) of the projective plane. In our example, this class contains two lines: $L_1$ consists of points 2 and 3, and $L_2$ consists of points 1 and 4. The remaining classes of the net are formed in a corresponding manner from the other points of the first line of the projective plane.

If we use all $(n + 1)$ classes to specify KWTA constraints, the nodes are over-constrained and the network has no stable states. We can obtain $n$ different codes by using $1, 2, \ldots,$ up to $n$ classes to specify constraints. (The code constructed with two classes is identical to the two-dimensional code in Section 2.1!) Experimentally, we have found that these decoders perform near-optimal decoding on their corresponding code. A code constructed with $i$ nets has a minimum distance of at least $2i$. Thus, a code of size $n^2$ (i.e., the codewords contain $n^2$ bits) can be constructed

with minimum distance up to $2n$. The rate of these codes in general can be bounded only roughly.

We found that we could embed the decoder with $a$ nets in an integrated circuit with width proportional to $\frac{1}{2}an^3$, or area proportional to the cube of the number of processors. In a typical VLSI process, one could implement systems with 484 ($a = 2$, $n = 22$), 81 ($a = 3$, $n = 9$) or 64 ($a = 4$, $n = 8$) nodes.

# 3   SUMMARY

We have simulated and analyzed analog neural networks which perform near-optimal decoding of certain families of nonlinear codes. Furthermore, we have shown that nontrivial implementations could be constructed. This work is discussed in more detail in [Erlanson, 1991].

**References**

E. Majani, R. Erlanson and Y.S. Abu-Mostafa, "On the K-Winners-Take-All Feedback Network," *Advances in Neural Information Processing Systems*, D. Touretzky (ed.), Vol. 1, pp. 634–642, 1989.

R. Erlanson and Y.S. Abu-Mostafa, "Using an Analog Neural Network for Decoding," *Proceedings of the 1988 Connectionist Models Summer School*, D. Touretzky, G. Hinton, T. Sejnowski (eds.), pp. 186–190, 1988.

J.C. Platt and J.J. Hopfield, "Analog decoding using neural networks," *AIP Conference Proceedings #151, Neural Networks for Computing*, J. Denker (ed.), pp. 364-369, 1986.

R. Erlanson, "Soft-Decision Decoding of a Family of Nonlinear Codes Using a Neural Network," PhD. Thesis, California Institute of Technology, 1991.

Table 2: Constructing a Net from a Projective Plane.

| projective plane's points: | 1 | 2 | 3 | 4 | 5 | 6 | 7 | |
|---|---|---|---|---|---|---|---|---|
| | 1 | 1 | | 1 | | | | |
| | | 1 | 1 | | 1 | | | |
| | | | 1 | 1 | | 1 | | |
| lines: | | | | 1 | 1 | | 1 | |
| | 1 | | | | 1 | 1 | | $L_1$ |
| | | 1 | | | | 1 | 1 | |
| | 1 | | 1 | | | | 1 | $L_2$ |
| net's points: | | | | 1 | 2 | 3 | 4 | |


## Footnotes

*currently at: Hughes Network Systems, 10790 Roselle St., San Diego, CA 92121
